# Practical Large-Scale Optimization
# for Max-Norm Regularization

**Jason Lee**
Institute of Computational and Mathematical Engineering
Stanford University
email: jl115@yahoo.com

**Benjamin Recht**
Department of Computer Sciences
University of Wisconsin-Madison
email: brecht@cs.wisc.edu

**Ruslan Salakhutdinov**
Brain and Cognitive Sciences and CSAIL
Massachusetts Institute of Technology
email: rsalakhu@mit.edu

**Nathan Srebro**
Toyota Technological Institute at Chicago
email: nati@ttic.edu

**Joel A. Tropp**
Computing and Mathematical Sciences
California Institute of Technology
email: jtropp@acm.caltech.edu

## Abstract

The *max-norm* was proposed as a convex matrix regularizer in [1] and was shown to be empirically superior to the trace-norm for collaborative filtering problems. Although the max-norm can be computed in polynomial time, there are currently no practical algorithms for solving large-scale optimization problems that incorporate the max-norm. The present work uses a factorization technique of Burer and Monteiro [2] to devise scalable first-order algorithms for convex programs involving the max-norm. These algorithms are applied to solve huge collaborative filtering, graph cut, and clustering problems. Empirically, the new methods outperform mature techniques from all three areas.

## 1 Introduction

A foundational concept in modern machine learning is to construct models for data by balancing the complexity of the model against fidelity to the measurements. In a wide variety of applications, such as collaborative filtering, multi-task learning, multi-class learning and clustering of multivariate observations, matrices offer a natural way to tabulate data. For such matrix models, the matrix rank provides an intellectually appealing way to describe complexity. The intuition behind this approach holds that many types of data arise from a noisy superposition of a small number of simple (i.e., rank-one) factors.

Unfortunately, optimization problems involving rank constraints are computationally intractable except in a few basic cases. To address this challenge, researchers have searched for alternative complexity measures that can also promote low-rank models. A particular example of a low-rank regularizer that has received a huge amount of recent attention is the *trace-norm*, equal to the sum of the matrix's singular values (See the comprehensive survey [3] and its bibliography). The trace-norm promotes low-rank decompositions because it minimizes the $\ell_1$ norm of the vector of singular values, which encourages many zero singular values.

Although the trace-norm is a very successful regularizer in many applications, it does not seem to be widely known or appreciated that there are many other interesting norms that promote low rank. The

paper [4] is one of the few articles in the machine learning literature that pursues this idea with any vigor. The current work focuses on another rank-promoting regularizer, sometimes called the *max-norm*, that has been proposed as an alternative to the rank for collaborative filtering problems [1, 5]. The max-norm can be defined via matrix factorizations:

$$\|\boldsymbol{X}\|_{\max} := \inf \left\{ \|\boldsymbol{U}\|_{2,\infty} \|\boldsymbol{V}\|_{2,\infty} : \boldsymbol{X} = \boldsymbol{U}\boldsymbol{V}' \right\} \tag{1}$$

where $\|\cdot\|_{2,\infty}$ denotes the maximum $\ell_2$ row norm of a matrix:

$$\|\boldsymbol{A}\|_{2,\infty} := \max_j \left( \sum_k \boldsymbol{A}_{jk}^2 \right)^{1/2}.$$

For general matrices, the computation of the max-norm can be rephrased as a semidefinite program; see (4) below. When $\boldsymbol{X}$ is positive semidefinite, we may force $\boldsymbol{U} = \boldsymbol{V}$ and then verify that $\|\boldsymbol{X}\|_{\max} = \max_j x_{jj}$, which should explain the terminology.

The fundamental result in the metric theory of tensor products, due to Grothendieck, states that the max-norm is comparable with a nuclear norm (see Chapter 10 of [6]):

$$\|\boldsymbol{X}\|_{\max} \approx \inf \left\{ \|\boldsymbol{\sigma}\|_1 : \boldsymbol{X} = \sum_j \sigma_j \boldsymbol{u}_j \boldsymbol{v}_j' \text{ where } \|\boldsymbol{u}_j\|_\infty = 1 \text{ and } \|\boldsymbol{v}_j\|_\infty = 1 \right\}.$$

The factor of equivalence $1.676 \leq \kappa_{\mathrm{G}} \leq 1.783$ is called Grothendieck's constant. The trace-norm, on the other hand, is equal to

$$\|\boldsymbol{X}\|_{\mathrm{tr}} := \inf \left\{ \|\boldsymbol{\sigma}\|_1 : \boldsymbol{X} = \sum_j \sigma_j \boldsymbol{u}_j \boldsymbol{v}_j' \text{ where } \|\boldsymbol{u}_j\|_2 = 1 \text{ and } \|\boldsymbol{v}_j\|_2 = 1 \right\}.$$

This perspective reveals that the max-norm promotes low-rank decompositions with factors in $\ell_\infty$, rather than the $\ell_2$ factors produced by the trace-norm! Heuristically, we expect max-norm regularization to be effective for uniformly bounded data, such as preferences.

The literature already contains theoretical and empirical evidence that the max-norm is superior to the trace-norm for certain types of problems. Indeed, the max-norm offers better generalization error bounds for collaborative filtering [5], and it outperforms the trace-norm in small-scale experiments [1]. The paper [7] provides further evidence that the max-norm serves better for collaborative filtering with nonuniform sampling patterns.

We believe that the max-norm has not achieved the same prominence as the trace-norm because of an apprehension that it is challenging to solve optimization problems involving a max-norm regularizer. The goal of this paper is to refute this misconception.

We provide several algorithms that are effective for very large scale problems, and we demonstrate the power of the max-norm regularizer using examples from a variety of applications. In particular, we study convex programs of the form

$$\min f(\boldsymbol{X}) + \mu \|\boldsymbol{X}\|_{\max} \tag{2}$$

where $f$ is a smooth function and $\mu$ is a positive penalty parameter. Section 4 outlines a proximal-point method, based on the work of Fukushima and Mine [8], for approaching (2). We also study the bound-constrained problem

$$\min f(\boldsymbol{X}) \quad \text{subject to} \quad \|\boldsymbol{X}\|_{\max} \leq B. \tag{3}$$

Of course, (2) and (3) are equivalent for appropriate choices of $\mu$ and $B$, but we describe scenarios where there may be a preference for one versus the other. Section 3 provides a projected gradient method for (3), and Section 5 develops a stochastic implementation that is appropriate for decomposable loss functions. These methods can be coded up in a few lines of numerical python or Matlab, and they scale to huge instances, even on a standard desktop machine. In Section 6, we apply these new algorithms to large-scale collaborative filtering problems, and we demonstrate performance superior to methods based on the trace-norm. We apply the algorithms to solve enormous instances of graph cut problems, and we establish that clustering based on these cuts outperforms spectral clustering on several data sets.

## 2 The SDP and Factorization Approaches

The max-norm of an $m \times n$ matrix $\boldsymbol{X}$ can be expressed as the solution to a semidefinite program:

$$\|\boldsymbol{X}\|_{\max} = \min t \quad \text{subject to} \quad \begin{bmatrix} \boldsymbol{W}_1 & \boldsymbol{X} \\ \boldsymbol{X}' & \boldsymbol{W}_2 \end{bmatrix} \succeq \boldsymbol{0}, \quad \operatorname{diag}(\boldsymbol{W}_1) \leq t, \quad \operatorname{diag}(\boldsymbol{W}_2) \leq t. \quad (4)$$

Unfortunately, standard interior-point methods for this problem do not scale to matrices with more than a few hundred rows or columns. For large-scale problems, we use an alternative formulation suggested by (1) that explicitly works with a factorization of the decision variable $\boldsymbol{X}$.

We employ an idea of Burer and Monteiro [2] that has far reaching consequences. The positive definite constraint in the SDP formulation above is trivially satisfied if we define $\boldsymbol{L}$ and $\boldsymbol{R}$ via

$$\begin{bmatrix} \boldsymbol{W}_1 & \boldsymbol{X} \\ \boldsymbol{X}' & \boldsymbol{W}_2 \end{bmatrix} = \begin{bmatrix} \boldsymbol{L} \\ \boldsymbol{R} \end{bmatrix} \begin{bmatrix} \boldsymbol{L} \\ \boldsymbol{R} \end{bmatrix}'.$$

Burer and Monteiro showed that as long as $\boldsymbol{L}$ and $\boldsymbol{R}$ have sufficiently many columns, then the global optimum of (4) is equal to that of

$$\|\boldsymbol{X}\|_{\max} = \min_{(\boldsymbol{L}, \boldsymbol{R}) \,:\, \boldsymbol{L}\boldsymbol{R}' = \boldsymbol{X}} \max\{\|\boldsymbol{L}\|_{2,\infty}^2, \|\boldsymbol{R}\|_{2,\infty}^2\}. \quad (5)$$

In particular, we may assume that the number of columns is less than $m + n$. This formulation of the max-norm is nonconvex because it involves a constraint on the product $\boldsymbol{L}\boldsymbol{R}'$, but Burer and Monteiro proved that each local minimum of the reformulated problem is also a global optimum [9]. If we select $\boldsymbol{L}$ and $\boldsymbol{R}$ to have a very small number of columns, say $r$, then the number of real decision variables in the optimization problems (2) and (3) is reduced from $mn$ to $r(m + n)$, a dramatic improvement in the dimensionality of the problem. On the other hand, the new formulation is nonconvex with respect to $\boldsymbol{L}$ and $\boldsymbol{R}$ so it might not be efficiently solvable. In what follows, we present fast, first-order methods for solving (2) and (3) via this low-dimensional factored representation.

## 3 Projected Gradient Method

The constrained formulation (3) admits a simple projected gradient algorithm. We replace $\boldsymbol{X}$ with the product $\boldsymbol{L}\boldsymbol{R}'$ and use the factored form of the max-norm (5) to obtain

$$\text{minimize}_{(\boldsymbol{L}, \boldsymbol{R})} f(\boldsymbol{L}\boldsymbol{R}') \quad \text{subject to } \max\{\|\boldsymbol{L}\|_{2,\infty}^2, \|\boldsymbol{R}\|_{2,\infty}^2\} \leq B. \quad (6)$$

The projected gradient descent method fixes a step size $\tau$ and computes updates with the rule

$$\begin{bmatrix} \boldsymbol{L} \\ \boldsymbol{R} \end{bmatrix} \leftarrow \mathcal{P}_B \left( \begin{bmatrix} \boldsymbol{L} - \tau \nabla f(\boldsymbol{L}\boldsymbol{R})\boldsymbol{R} \\ \boldsymbol{R} - \tau \nabla f(\boldsymbol{L}\boldsymbol{R})'\boldsymbol{L} \end{bmatrix} \right)$$

where $\mathcal{P}_B$ denotes the Euclidean projection onto the set $\{(\boldsymbol{L}, \boldsymbol{R}) : \max(\|\boldsymbol{L}\|_{2,\infty}^2, \|\boldsymbol{R}\|_{2,\infty}^2) \leq B\}$. This projection can be computed by re-scaling the rows of the current iterate whose norms exceed $\sqrt{B}$ so their norms equal $\sqrt{B}$. Rows with norms less than $\sqrt{B}$ are unchanged by the projection. The projected gradient algorithm is elegant and simple, and it has an online implementation, described below. Moreover, using an Armijo line search rule to guarantee sufficient decrease of the cost function, we can guarantee convergence to a stationary point of (3); see [10, Sec. 2.3].

## 4 Proximal Point Method for Penalty Formulation

Solving (2) is slightly more complicated than its constrained counterpart. We employ a classical proximal point method, proposed by Fukushima and Mine [8], which forms the algorithmic foundation of many popular first-order methods of for $\ell_1$-norm minimization [11, 12] and trace-norm minimization [13, 14]. The key idea is that our cost function is the sum of a smooth term plus a convex term. At each iteration, we replace the smooth term by a linear approximation. The new cost function can then be minimized in closed form. Before describing the proximal point algorithm in detail, we first discuss how a simple max-norm problem (the Frobenius norm plus a max-norm penalty) admits an explicit formula for its unique optimal solution.

Consider the simple regularization problem

$$\text{minimize}_{\boldsymbol{W}} \quad \|\boldsymbol{W} - \boldsymbol{V}\|_F^2 + \beta \|\boldsymbol{W}\|_{2,\infty}^2 \quad (7)$$

---

**Algorithm 1** Compute $\boldsymbol{W} = \mathrm{squash}(\boldsymbol{V}, \beta)$

---

**Require:** A $d \times D$ matrix $\boldsymbol{V}$, a positive scalar $\beta$.
**Ensure:** A $d \times D$ matrix $\boldsymbol{W} \in \arg\min_{\boldsymbol{Z}} \quad \|\boldsymbol{Z} - \boldsymbol{V}\|_F^2 + \beta \|\boldsymbol{Z}\|_{2,\infty}^2$.

1: **for** $k = 1$ to $d$ **set** $n_k \leftarrow \|\boldsymbol{v}_k\|_2$
2: **sort** $\{n_k\}$ in descending order. Let $\pi$ denote the sorting permutation such that $n_{\pi(j)}$ is the $j$th largest element in the sequence.
3: **for** $k = 1$ to $d$ **set** $s_k \leftarrow \sum_{i=1}^{k} n_{\pi(i)}$.
4: $q \leftarrow \max\{k \ : \ n_{\pi(k)} \geq \frac{s_k}{k+\beta}\}$
5: $\eta \leftarrow \frac{s_q}{q+\beta}$
6: **for** $k = 1$ to $d$, **if** $k \leq q$, **set** $\boldsymbol{w}_{\pi(k)} \leftarrow \eta \boldsymbol{v}_{\pi(k)}/\|\boldsymbol{v}_{\pi(k)}\|_2$. **otherwise set** $\boldsymbol{w}_{\pi(k)} \leftarrow \boldsymbol{v}_{\pi(k)}$

---

where $\boldsymbol{W}$ and $\boldsymbol{V}$ are $d \times D$ matrices. Just as with $\ell_1$-norm and trace-norm regularization, this problem can be solved in closed form. An efficient algorithm to solve (7) is given by Algorithm 1. We call this procedure squash because the rows of $\boldsymbol{V}$ with large norm have their magnitude clipped at a critical value $\eta = \eta(\boldsymbol{V}, \beta)$.

**Proposition 4.1** $\mathrm{squash}(\boldsymbol{V}, \beta)$ *is an optimal solution of (7)*

The proof of this proposition follows from an analysis of the KKT conditions for the regularized problem. We include a full derivation in the appendix. Note that squash can be computed in $O(d \max\{\log(d), D\})$ flops. Computing the row norms requires $O(dD)$ flops, and then the sort requires $O(d \log d)$ flops. Computing $\eta$ and $q$ require $O(d)$ operations. Constructing $\boldsymbol{W}$ then requires $O(dD)$ operations.

With the squash function in hand, we can now describe our proximal-point algorithm. Replace the decision variable $\boldsymbol{X}$ in (2) with $\boldsymbol{L}\boldsymbol{R}'$. With this substitution and the factored form of the max-norm, (5), Problem (2) reduces to

$$\mathrm{minimize}_{(\boldsymbol{L},\boldsymbol{R})} f(\boldsymbol{L}\boldsymbol{R}') + \mu \max\{\|\boldsymbol{L}\|_{2,\infty}^2, \|\boldsymbol{R}\|_{2,\infty}^2\} . \tag{8}$$

For ease of notation, define $\boldsymbol{A}$ to be the matrix of factors stacked on top of one another $\boldsymbol{A} = \begin{bmatrix} \boldsymbol{L} \\ \boldsymbol{R} \end{bmatrix}$. With this notation, we have $\|\boldsymbol{A}\|_{2,\infty}^2 = \max\{\|\boldsymbol{L}\|_{2,\infty}^2, \|\boldsymbol{R}\|_{2,\infty}^2\}$. Also let $\tilde{f}(\boldsymbol{A})$ denote $f(\boldsymbol{L}\boldsymbol{R}')$, and $\varphi(\boldsymbol{A}) := \tilde{f}(\boldsymbol{A}) + \mu \|\boldsymbol{A}\|_{2,\infty}^2$.

Using the squash algorithm, we can solve

$$\mathrm{minimize} \langle \nabla \tilde{f}(\boldsymbol{A}_k), \boldsymbol{A} \rangle + \tau_k^{-1} \|\boldsymbol{A} - \boldsymbol{A}_k\|_F^2 + \mu \|\boldsymbol{A}\|_{2,\infty}^2 \tag{9}$$

in closed form. To see this, complete the square and multiply by $\tau_k$. Then (9) is equivalent to (7) with the identifications $\boldsymbol{W} = \boldsymbol{A}$, $\boldsymbol{V} = \boldsymbol{A}_k - \tau_k \nabla \tilde{f}(\boldsymbol{A}_k)$, $\beta = \tau_k \mu$. That is, the optimal solution of (9) is $\mathrm{squash}\left(\boldsymbol{A}_k - \tau_k \nabla \tilde{f}(\boldsymbol{A}_k), \tau_k \mu\right)$.

We can now directly apply the proximal-point algorithm of Fukushima and Mine, detailed in Algorithm 2. Step 2 is the standard linearized proximal-point method that is prevalent in convex algorithms like Mirror Descent and Nesterov's optimal method. The cost function $\tilde{f}$ is replaced with a quadratic approximation localized at the previous iterate $\boldsymbol{A}_k$, and the resulting approximation (9) can be solved in closed form. Step 3 is a backtracking line search that looks for a step that obeys an Armijo step rule. This linesearch guarantees that the algorithm produces a sufficiently large decrease of the cost function at each iteration, but it may require several function evaluations to find $l$. This algorithm is guaranteed to converge to a critical point of (8) as long as the step sizes are chosen commensurate with the norm of the Hessian [8]. In particular, Nesterov has recently shown that if $\tilde{f}$ has a Lipschitz-continuous gradient with Lipschitz constant $L$, then the algorithm will converge at a rate of $1/k$ where $k$ is the iteration counter [15].

---

**Algorithm 2** A proximal-point method for max-norm regularization

---

**Require:** Algorithm parameters $\alpha > 0$, $1 > \gamma > 0$, $\epsilon_{\text{tol}} > 0$. A sequence of positive numbers $\{\tau_k\}$. An initial point $\boldsymbol{A}_0 = (\boldsymbol{L}_0, \boldsymbol{R}_0)$ and a counter $k$ set to 0.

**Ensure:** A critical point of (8).

1: **repeat**

2:　　Solve (9) to find $\hat{\boldsymbol{A}}_k$. That is, $\hat{\boldsymbol{A}}_k \leftarrow \text{squash}\left(\boldsymbol{A}_k - \tau_k \nabla \tilde{f}(\boldsymbol{A}_k), \tau_k \mu\right)$.

3:　　Compute the smallest nonnegative integer $l$ such that

$$\varphi(\boldsymbol{A}_k + \gamma^l(\hat{\boldsymbol{A}}_k - \boldsymbol{A}_k)) \leq \varphi(\boldsymbol{A}_k) - \alpha\gamma^l \|\boldsymbol{A}_k - \hat{\boldsymbol{A}}_k\|_F^2 \, .$$

4:　　**set** $\boldsymbol{A}_{k+1} \leftarrow (1 - \gamma^l)\boldsymbol{A}_k + \gamma^l \hat{\boldsymbol{A}}_k, \;\; k \leftarrow k + 1$.

5: **until** $\frac{\|\boldsymbol{A}_k - \hat{\boldsymbol{A}}_k\|_F^2}{\|\boldsymbol{A}_k\|_F^2} < \epsilon_{\text{tol}}$

---

## 5 Stochastic Gradient

For many problems, including matrix completion and max-cut problems, the cost function decomposes over the individual entries in the matrix, so the function $f(\boldsymbol{LR}')$ takes the particularly simple form:

$$f(\boldsymbol{L}, \boldsymbol{R}) = \sum_{i,j \in S} \ell(Y_{ij}, \boldsymbol{L}_i' \boldsymbol{R}_j) \tag{10}$$

where $\ell$ is some fixed loss function, $S$ is a set of row-column indices, $Y_{ij}$ are some real numbers, and $\boldsymbol{L}_i$ and $\boldsymbol{R}_j$ denote the $i$th row of $\boldsymbol{L}$ and $j$th row of $\boldsymbol{R}$ respectively. When dealing with very large datasets, $S$ may consist of hundreds of millions of pairs, and there are algorithmic advantages to utilizing stochastic gradient methods that only query a very small subset of $S$ at each iteration. Indeed, the above decomposition for $f$ immediately suggests a stochastic gradient method: pick one training pair $(i, j)$ at random at each iteration, take a step in the direction opposite the gradient of $\ell(Y_{i,j}, \boldsymbol{L}_i' \boldsymbol{R}_j)$ and then either apply the projection $\mathcal{P}_B$ described in Section 3 or the squash function described in 4.

The projection $\mathcal{P}_B$ is particularly easy to compute in the stochastic setting. Namely, if $\|\boldsymbol{L}_i\|^2 > B$, we project it back so that $\|\boldsymbol{L}_i\| = \sqrt{B}$, otherwise we do not do anything (and similarly for $\boldsymbol{R}_j$). We need not look at any other rows of $\boldsymbol{L}$ and $\boldsymbol{R}$. As we demonstrate in experimental results section, this simple algorithm is computationally as efficient as optimization with the trace-norm.

We can also implement an efficient algorithm for stochastic gradient descent for problem (2). If we wanted to apply the squash algorithm to such a stochastic gradient step, only the norms corresponding to $\boldsymbol{L}_i$ and $\boldsymbol{R}_j$ would be modified. Hence, in Algorithm 1, if the set of row norms of $\boldsymbol{L}$ and $\boldsymbol{R}$ is sorted from the previous iteration, we can implement a balanced-tree data structure that allows us to perform individual updates in amortized logarithmic time. We leave such an implementation to future work. In the experiments, however, we demonstrate that the proximal point method is still quite efficient and fast when dealing with stochastic gradient updates corresponding to medium-size batches $\{(i, j)\}$ selected from $S$, even if a full sort is performed at each squash operation.

## 6 Numerical Experiments

**Matrix Completion.** We tested our proximal point and projected gradient methods on the Netflix dataset, which is the largest publicly available collaborative filtering dataset. The training set contains 100,480,507 ratings from 480,189 anonymous users on 17,770 movie titles. Netflix also provides a qualification set, containing 1,408,395 ratings. The "qualification set" pairs were selected by Netflix from the most recent ratings for a subset of the users. As a baseline, Netflix provided the test score of its own system trained on the same data, which is 0.9514. This dataset is interesting for several reasons. First, it is very large, and very sparse (98.8% sparse). Second, the dataset is very imbalanced, with highly nonuniform samples. It includes users with over 10,000 ratings as well as users who rated fewer than 5 movies.

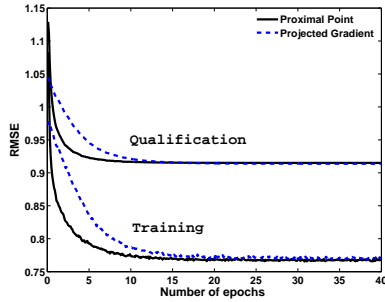

| Algorithm | $f(\boldsymbol{X})$ | Training RMSE | | Qual |
| | | $\|\boldsymbol{X}\|_{\max}$ | $f(\boldsymbol{X})+$ $+\mu\,\|\boldsymbol{X}\|_{\max}$ | $f(\boldsymbol{X})$ |
| --- | --- | --- | --- | --- |
| Proximal Point | 0.7676 | 2.5549 | 0.7689 | 0.9150 |
| Projected Gradient | 0.7728 | 2.2500 | 0.7739 | 0.9138 |
| Trace-norm | - | - | - | 0.9235 |
| Weighted Trace-norm | - | - | - | 0.9105 |

Figure 1: *Performance of regularization methods on the Netflix dataset.*

For the netflix dataset, we will evaluate our algorithms based on the root mean squared error (RMSE) of their predictions. To this end, the objective we seek to minimize takes the following form:

$$\text{minimize}_{\boldsymbol{L},\boldsymbol{R}} \ \frac{1}{|S|} \sum_{(i,j)\in S} (Y_{ij} - \boldsymbol{L}_i'\boldsymbol{R}_j)^2 + \mu \max\{\|\boldsymbol{L}\|_{2,\infty}^2, \|\boldsymbol{R}\|_{2,\infty}^2\}$$

where $S$ here represents the set of observed user-movie pairs and $Y_{ij}$ denote the provided ratings. For all of our experiments, we learned a factorization $\boldsymbol{L}'\boldsymbol{R}$ with $k = 30$ dimensions (factors).

In our experiments, all ratings were normalized to be zero-mean by subtracting 3.6. To speed up learning, we subdivided the Netflix dataset into minibatches, each containing 100,000 user/movie/rating triplets. Both proximal-point and projected gradient methods performed 40 epochs (or passes through the training set), with parameters $\{\boldsymbol{L}, \boldsymbol{R}\}$ updated after each minibatch. For both algorithms we used momentum of 0.9, and a step size of 0.005, which was decreased by a factor of 0.8 after each epoch. For the proximal-point method, $\mu$ was set to $5\times10^{-4}$, and for the projected gradient algorithm, $B$ was set to 2.25. The running times of both algorithms on this large-scale Netflix dataset is comparable. On a 2.5 GHz Intel Xeon, our implementation of projected gradient takes 20.1 minutes per epoch, whereas the proximal-point method takes about 19.6 minutes.

Figure 1 shows predictive performance of both the proximal-point and projected gradient algorithms on the training and qualification set. Observe that the proximal-point algorithm converges considerably faster than projected gradient, but both algorithms achieve a similar RMSE of 0.9150 (proximal point) and 0.9138 (projected gradient) on the qualification set. Figure 1, left panel, further shows that the max-norm based regularization significantly outperforms the corresponding trace-norm based regularization, which is widely used in many large-scale collaborative filtering applications. We also note that the differences between the max-norm and the weighted trace-norm [7] are rather small, with the weighted trace-norm slightly outperforming max-norm.

**Gset Max-Cut Experiments.** In the MAX-CUT problem, we are given a graph $G = (V, E)$, and we aim to solve the problem

$$\text{minimize} \sum_{(i,j)\in E} (1 - x_i x_j) \text{ subject to } x_i^2 = 1 \ \forall i \in V$$

The heralded Goemans-Williamson relaxation [16] converts this problem into a constrained, symmetric max-norm problem:

$$\text{minimize} \sum_{(i,j)\in E} (1 - X_{ij}) \text{ subject to } \|\boldsymbol{X}\|_{\max} \le 1, \ \boldsymbol{X} \succeq 0.$$

In our nonconvex formulation, this optimization becomes

$$\text{minimize} \sum_{(i,j)\in E} (1 - \boldsymbol{A}_i'\boldsymbol{A}_j) \text{ subject to } \|\boldsymbol{A}\|_{2,\infty}^2 \le 1.$$

Since the decision variable is symmetric and positive definite, we only need one factor $\boldsymbol{A}$ of size $|V| \times r$. In all of our experiments with MAX-CUT type problems, we fixed $r = 20$. We used a diminishing step size rule of $\tau_k = \frac{\tau_0}{\sqrt{k}}$ where $k$ is the iteration counter.

| | Primal Obj. | Time (.1%) | Iterations (.1%) | Time (1%) | Iterations (1%) | SDPLR Obj. | SDPLR Time | $|V|$ | $|E|$ |
|---|---|---|---|---|---|---|---|---|---|
| G22 | 14128.5 | 0.6 | 150 | 0.4 | 100 | 14135.7 | 3 | 2000 | 19990 |
| G35 | 8007.4 | 0.5 | 200 | 0.3 | 100 | 8014.6 | 4 | 2000 | 11778 |
| G36 | 7998.3 | 0.5 | 200 | 0.3 | 100 | 8005.9 | 7 | 2000 | 11766 |
| G58 | 20116.6 | 2 | 300 | .7 | 100 | 20135.90 | 29 | 5000 | 29570 |
| G60 | 15207.0 | 2.1 | 400 | 0.29 | 50 | 15221.9 | 6 | 7000 | 17148 |
| G67 | 7736.4 | 21.4 | 2050 | 1.3 | 100 | 7744.1 | 15 | 10000 | 20000 |
| G70 | 9851.51 | 8.7 | 1700 | .5 | 100 | 9861.2 | 21 | 10000 | 9999 |
| G72 | 7800.4 | 13.8 | 2250 | .6 | 100 | 7808.2 | 15 | 10000 | 20000 |
| G77 | 11034.1 | 18.6 | 2150 | .9 | 100 | 11045.1 | 20 | 14000 | 28000 |
| G81 | 15639.6 | 28.4 | 2200 | 1.35 | 100 | 15655.2 | 33 | 20000 | 40000 |

Table 1: *Performance of projected gradient on Gset graphs.* Columns show primal objective within .1% of optimal, running time for .1% of optimal, number of iterations to reach .1% of optimal, running time for 1% of optimal, number of iterations to reach 1% of optimal, primal objective using SDPLR, running time of SDPLR, number of vertices, and number of edges. In our experiments, we set $\tau_0 = 1$.

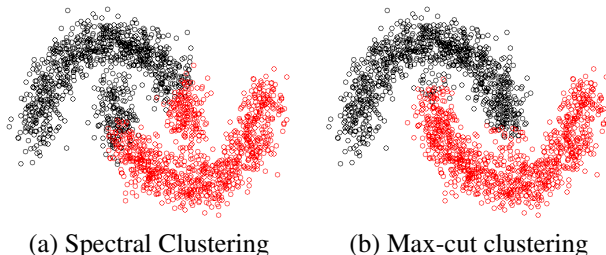

(a) Spectral Clustering  (b) Max-cut clustering

Figure 2: *Comparison of spectral clustering (left) with* MAX-CUT *clustering (right).*

We tested our projected gradient algorithm on graphs drawn from the Gset, a collection of graphs designed for testing the efficacy of max-cut algorithms [17]. The results for a subset of these appears in Table 1 along with a comparison against a C implementation of Burer's SDPLR code which has been optimized for the particular structure of the MAX-CUT problem [18]. On the same modern hardware, a Matlab implementation of our projected gradient method can reach .1% of the optimal value faster than the optimized and compiled SDPLR code.

**2-class Clustering Experiments.** For the 2-class clustering problem, we first build a $K$-nearest neighbor graph with $K = 10$ and weights $w_{ij}$ defined as $w_{ij} = \max(s_i(j), s_j(i))$, with $s_i(j) = \exp\left(-\frac{||x_i - x_j||^2}{2\sigma_i^2}\right)$ and $\sigma_i$ equal to the distance from $x_i$ to its $K$th closest neighbor. We then choose a scalar $\delta > 0$ and define an inverse similarity adjacency matrix $\boldsymbol{Q}$ by $Q_{ij} = \delta - W_{ij}$. The parameter $\delta$ controls the balancing of the clusters, a large value of $\delta$ forces the clusters to be of equal size. We solve the MAX-CUT problem on the graph $\boldsymbol{Q}$ to find our cluster assignments.

As a synthetic example, we generated a "two moons" dataset consisting of two half-circles in $\mathbb{R}^2$ with the bottom half circle shifted to the right by 1/2 and shifted up by 1/2. The data is then embedded into $\mathbb{R}^D$ and each embedded component is corrupted with Gaussian noise with variance $\sigma^2$. For the two moons experiments, we fix $D = 100$, $n = 2000$ and $\sigma = \sqrt{.02}$ as done in [19]. The parameters are set to $\delta = .01$ and $\tau_0 = 3/2$; the algorithm was executed for $1500$ iterations. For the clustering experiments, we repeat the randomized rounding technique [16] for $100$ trials, and we choose the rounding with highest primal objective.

We compare our MAX-CUT clusterings with the spectral clustering method [20] and the Total Variation Graph Cut algorithm [19]. Figure 2 shows the clustering results for spectral clustering and maxcut clustering. In all the trials, spectral clustering incorrectly clustered the two ends of both half-circles. For the clustering problems, the two measures of performance we consider are misclassification error rate (number of misclassified points divided by $n$) and cut cost. The cut cost is defined as $\sum_{i \in V_1, j \in V_2} W_{ij}$. The MAX-CUT clustering obtained smaller misclassification error in 98 of the 100 trials we performed and smaller cut cost in every trial.

On the MNIST database, we build the 10-NN graph described above on the digits 4 and 9, where we set $\delta = .001$ and $r = 8$. The NN-graph is of size $14,000$ and the MAX-CUT algorithm takes

| | max-cut | | | | spectral | | TV |
|---|---|---|---|---|---|---|---|
| | **Error Rate** | **Cost** | **Time** | $\frac{\min(|V_1|,|V_2|)}{|V_1|+|V_2|}$ | **Error Rate** | **Cost** | **Error Rate** |
| **Two Moons** | 0.053 | 311.9 | 13 | .495 | 0.171 | 387.8 | 0.082 |
| **MNIST 4 and 9** | 0.021 | 1025.5 | 90 | .493 | 0.458 | 1486.5 | N/A |
| **MNIST 3 and 5** | 0.016 | 830.9 | 53 | .476 | 0.092 | 2555.1 | N/A |

Table 2: *Clustering results.* Error rate, cut cost, and running time comparison for MAX-CUT, spectral, and total variation (TV) algorithms. The balance of the cut is computed as $\frac{\min(|V_1|,|V_2|)}{|V_1|+|V_2|}$. The two moons results are averaged over 100 trials.

approximately 1 minute to run 1,000 iterations. The same procedure is repeated for the digits 3 and 5. The results are shown in Table 2. Our MAX-CUT clustering algorithm again performs substantially better than the spectral method.

## 7 Summary

In this paper we presented practical methods for solving very large scale optimization problems involving a max-norm constraint or regularizer. Using this approaches, we showed evidence that the max-norm can often be superior to established techniques such as trace-norm regularization and spectral clustering, supplementing previous evidence on small-scale problems. We hope that the increasing evidence of the utility of max-norm regularization, combined with the practical optimization techniques we present here, will reignite interest in using the max-norm for various machine learning applications.

### Acknowledgements

RS supported by NSERC, Shell, and NTT Communication Sciences Laboratory. JAT supported by ONR award N00014-08-1-0883, DARPA award N66001-08-1-2065, and AFOSR award FA9550-09-1-0643. JL thanks TTI Chicago for hosting him while this work was completed.

## A    Proof of the correctness of squash

Rewrite (7) as the constrained optimization

$$\text{minimize}_{\boldsymbol{W},t} \quad \sum_{i=1}^{d}\|\boldsymbol{w}_i - \boldsymbol{v}_i\|^2 + \beta t$$
$$\text{subject to} \quad \|\boldsymbol{w}_i\|^2 \le t \quad \text{for } 1 \le i \le d$$

Forming a Lagrangian with a vector of Lagrange multipliers $\boldsymbol{p} \ge 0$

$$\mathcal{L}(\boldsymbol{W}, t, \boldsymbol{p}) = \sum_{i=1}^{d}\|\boldsymbol{w}_i - \boldsymbol{v}_i\|^2 + \beta t + \sum_{i=1}^{d} p_i(\|\boldsymbol{w}_i\|^2 - t),$$

the KKT conditions for this problem thus read (a) $\boldsymbol{w}_i = \frac{1}{1+p_i}\boldsymbol{v}_i$, (b) $\boldsymbol{p} \ge 0$, (c) $\sum_{i=1}^{d} p_i = \beta$, (d) $\|\boldsymbol{w}_i\|^2 \le t$ for $1 \le i \le d$, (e) $p_i > 0 \implies \|\boldsymbol{w}_i\|^2 = t$, and (f) $\|\boldsymbol{w}_i\|^2 < t \implies p_i = 0$.

With our candidate $\boldsymbol{W} = \text{squash}(\boldsymbol{V}, \beta)$, we need only find $t$ and $\boldsymbol{p}$ to verify the optimality conditions. Let $\pi$ be as in Algorithm 1 and set $t = \eta^2$ and

$$p_k = \begin{cases} \frac{\|\boldsymbol{v}_k\|}{\eta} - 1 & 1 \le \pi(k) \le q \\ 0 & \text{otherwise} \end{cases}$$

This definition of $\boldsymbol{p}$ immediately gives (a). For (b), note that by the definition of $q$, $\|\boldsymbol{v}_k\| \ge \eta$ for $1 \le \pi(k) \le q$. Thus, $\boldsymbol{p} \ge 0$. Moreover,

$$\sum_{k=1}^{d} p_k = \frac{\sum_{1 \le \pi(k) \le q}\|\boldsymbol{v}_k\|}{\eta} - q = q + \beta - q = \beta,$$

yielding (c). Also, by construction, $\|\boldsymbol{w}_k\| = \eta$ if $\pi(k) \le q$ verifying (e). Finally, again by the definition of $q$, we have

$$\|\boldsymbol{v}_{\pi(q+1)}\| < \frac{1}{\beta+q+1}\sum_{k=1}^{q+1}\|\boldsymbol{v}_{\pi(k)}\| = \frac{1}{\beta+q+1}\|\boldsymbol{v}_{\pi(q+1)}\| + \frac{\beta+q}{\beta+q+1}\eta$$

which implies $\|\boldsymbol{v}_{\pi(q+1)}\| < \eta$. Since $\|\boldsymbol{v}_k\| \le \|\boldsymbol{v}_{\pi(q+1)}\|$ for $\pi(k) > q$, this gives (d) and the slackness condition (f).

# References

[1] Nathan Srebro, Jason Rennie, and Tommi Jaakkola. Maximum margin matrix factorization. In *Advances in Neural Information Processing Systems*, 2004.

[2] Samuel Burer and R. D. C. Monteiro. A nonlinear programming algorithm for solving semidefinite programs via low-rank factorization. *Mathematical Programming (Series B)*, 95:329–357, 2003.

[3] Benjamin Recht, Maryam Fazel, and Pablo Parrilo. Guaranteed minimum rank solutions of matrix equations via nuclear norm minimization. *SIAM Review*, 2007. To appear. Preprint Available at `http://pages.cs.wisc.edu/˜brecht/publications.html`.

[4] Francis R. Bach, Julien Marial, and Jean Ponce. Convex sparse matrix factorizations. Preprint available at `arxiv.org/abs/0812.1869`, 2008.

[5] Nathan Srebro and Adi Shraibman. Rank, trace-norm and max-norm. In *18th Annual Conference on Learning Theory (COLT)*, 2005.

[6] G. J. O. Jameson. *Summing and Nuclear Norms in Banach Space Theory*. Number 8 in London Mathematical Society Student Texts. Cambridge University Press, Cambridge, UK, 1987.

[7] Ruslan Salakhutdinov and Nathan Srebro. Collaborative filtering in a non-uniform world: Learning with the weighted trace norm. Preprint available at `arxiv.org/abs/1002.2780`, 2010.

[8] Masao Fukushima and Hisashi Mine. A generalized proximal point algorithm for certain non-convex minimization problems. *International Journal of Systems Science*, 12(8):989–1000, 1981.

[9] Samuel Burer and Changhui Choi. Computational enhancements in low-rank semidefinite programming. *Optimization Methods and Software*, 21(3):493–512, 2006.

[10] Dimitri P. Bertsekas. *Nonlinear Programming*. Athena Scientific, Belmont, MA, 2nd edition, 1999.

[11] T Hale, W Yin, and Y Zhang. A fixed-point continuation method for l 1-regularized minimization with applications to compressed sensing. *Dept. Computat. Appl. Math., Rice Univ., Houston, TX, Tech. Rep. TR07-07*, 2007.

[12] Stephen J. Wright, Robert Nowak, and Mário A. T. Figueiredo. Sparse reconstruction by separable approximation. Journal version, to appear in IEEE Transactions on Signal Processing. Preprint available at `http:http://www.optimization-online.org/DB_HTML/2007/10/1813.html`, 2007.

[13] Jian-Feng Cai, Emmanuel J. Candès, and Zuowei Shen. A singular value thresholding algorithm for matrix completion. To appear in *SIAM J. on Optimization*. Preprint available at `http://arxiv.org/abs/0810.3286`, 2008.

[14] Shiqian Ma, Donald Goldfarb, and Lifeng Chen. Fixed point and Bregman iterative methods for matrix rank minimization. Preprint available at `http://www.optimization-online.org/DB_HTML/2008/11/2151.html`, 2008.

[15] Yurii Nesterov. Gradient methods for minimizing composite objective function. To appear. Preprint Available at `http://www.optimization-online.org/DB_HTML/2007/09/1784.html`, September 2007.

[16] M. X. Goemans and D. P. Williamson. Improved approximation algorithms for maximum cut and satisfiability problems using semidefinite programming. *Journal of the ACM*, 42:1115–1145, 1995.

[17] The Gset is available for download at `http://www.stanford.edu/˜yyye/yyye/Gset/`.

[18] Samuel Burer. Sdplr. Software available at `http://dollar.biz.uiowa.edu/˜sburer/www/doku.php?id=software#sdplr`.

[19] Arthur Szlam and Xavier Bresson. A total variation-based graph clustering algorithm for cheeger ratio cuts. To appear in *ICML 2010*. Preprint available at `ftp://ftp.math.ucla.edu/pub/camreport/cam09-68.pdf`, 2010.

[20] Jianbo Shi and Jitendra Malik. Normalized cuts and image segmentation. *IEEE Transactions on Pattern Analysis and Machine Intelligence*, 22(8):888–905, 2000.

